# Comparing the Performance of Connectionist and Statistical Classifiers on an Image Segmentation Problem

**Sheri L. Gish    W. E. Blanz**
IBM Almaden Research Center
650 Harry Road
San Jose, CA 95120

## ABSTRACT

In the development of an image segmentation system for real time image processing applications, we apply the classical decision analysis paradigm by viewing image segmentation as a pixel classification task. We use supervised training to derive a classifier for our system from a set of examples of a particular pixel classification problem. In this study, we test the suitability of a connectionist method against two statistical methods, Gaussian maximum likelihood classifier and first, second, and third degree polynomial classifiers, for the solution of a "real world" image segmentation problem taken from combustion research. Classifiers are derived using all three methods, and the performance of all of the classifiers on the training data set as well as on 3 separate entire test images is measured.

## 1   Introduction

We are applying the trainable machine paradigm in our development of an image segmentation system to be used in real time image processing applications. We view image segmentation as a classical decision analysis task; each pixel in a scene is described by a set of measurements, and we use that set of measurements with a classifier of our choice to determine the region or object within a scene to which that pixel belongs. Performing image segmentation as a decision analysis task provides several advantages. We can exploit the inherent trainability found in decision

analysis systems [1] and use supervised training to derive a classifier from a set of examples of a particular pixel classification problem. Classifiers derived using the trainable machine paradigm will exhibit the property of generalization, and thus can be applied to data representing a set of problems similar to the example problem. In our pixel classification scheme, the classifier can be derived solely from the quantitative characteristics of the problem data. Our approach eliminates the dependency on qualitative characteristics of the problem data which often is characteristic of explicitly derived classification algorithms [2,3].

Classical decision analysis methods employ statistical techniques. We have compared a connectionist system to a set of alternative statistical methods on classification problems in which the classifier is derived using supervised training, and have found that the connectionist alternative is comparable, and in some cases preferable, to the statistical alternatives in terms of performance on problems of varying complexity [4]. That comparison study also analyzed the alternative methods in terms of cost of implementation of the solution architecture in digital LSI. In terms of our cost analysis, the connectionist architectures were much simpler to implement than the statistical architectures for the more complex classification problems; this property of the connectionist methods makes them very attractive implementation choices for systems requiring hardware implementations for difficult applications.

In this study, we evaluate the performance of a connectionist method and several statistical methods as the classifier component of our real time image segmentation system. The classification problem we use is a "real world" pixel classification task using images of the size (200 pixels by 200 pixels) and variable data quality typical of the problems a production system would be used to solve. We thus test the suitability of the connectionist method for incorporation in a system with the performance requirements of our system, as well as the feasibility of our exploiting the advantages the simple connectionist architectures provide for systems implemented in hardware.

## 2    Methods

### 2.1    The Image Segmentation System

The image segmentation system we use is described in [5], and summarized in Figure 1. The system is designed to perform low level image segmentation in real time; for production, the feature extraction and classifier system components are implemented in hardware. The classifier parameters are derived during the Training Phase. A user at a workstation outlines the regions or objects of interest in a training image. The system performs low level feature extraction on the training image, and the results of the feature extraction plus the input from the user are combined automatically by the system to form a training data set. The system then applies a supervised training method making use of the training data set in order to derive the coefficients for the classifier which can perform the pixel classification task. The feature extraction process is capable of computing 14 classes of features for each pixel; up to 10 features with the highest discriminatory power are used to

describe all of the pixels in the image. This selection of features is based only on an analysis of the results of the feature extraction process and is independent of the supervised learning paradigm being used to derive the classifier [6]. The identical feature extraction process is applied in both the Training and Running Phases for a particular image segmentation problem.

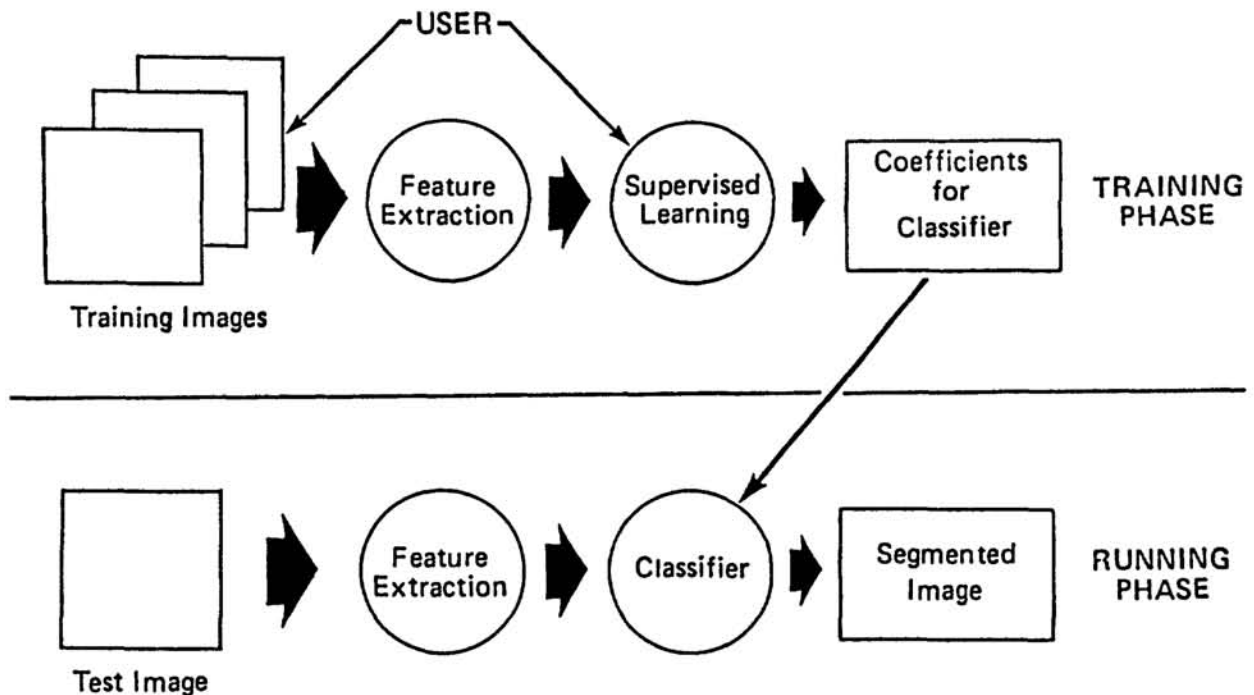

**Figure 1:** Diagram of the real time image segmentation system.

## 2.2   The Image Segmentation Problem

The image segmentation problem used in this study is from combustion research and is described in [3]. The images are from a series of images of a combustion chamber taken by a high speed camera during the inflammation process of a gas/air mixture. The segmentation task is to determine the area of inflamed gas in the image; therefore, the pixels in the image are classified into 3 different classes: cylinder, uninflamed gas, and flamed gas (See Figure 2). Exact determination of the area of flamed gas is not possible using pixel classification alone, but the greater the success of the pixel classification step, the greater the likelihood that a real time image segmentation system could be used successfully on this problem.

## 2.3   The Classifiers

The set of classifiers used in this study is composed of a connectionist classifier based on the Parallel Distributed Processing (PDP) model described in [7] and two statistical methods: a Gaussian maximum likelihood classifier (a Bayes classifier), and a polynomial classifier based on first, second, and third degree polynomials. This set of classifiers was used in a general study comparing the performance of

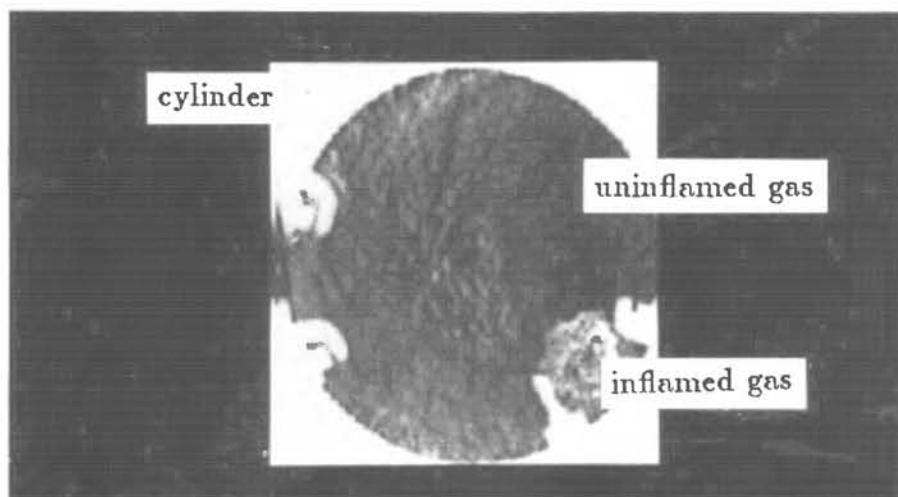

**Figure 2:** The image segmentation problem is to classify each image pixel into 1 of 3 regions.

the alternatives on a set of classification problems; all of the classifiers as well as adaptation procedures are described in detail in that study [4]. Implementation and adaptation of all classifiers in this study was performed as software simulation. The connectionist classifier was implemented in CMU Common Lisp running on an IBM RT workstation.

The connectionist classifier architecture is a multi-layer feedforward network with one hidden layer. The network is fully connected, but there are only connections between adjacent layers. The number of units in the input and output layers are determined by the number of features in the feature vector describing each pixel and a binary encoding scheme for the class to which the pixel belongs, respectively. The number of units in the hidden layer is an architectural "free parameter." The network used in this study has 10 units in the input layer, 12 units in the hidden layer, and 3 units in the output layer.

Network activation is achieved by using the continuous, nonlinear *logistic* function defined in [8]. The connectionist adaptation procedure is the application of the backpropagation learning rule also defined in [8]. For this problem, the learning rate $\eta = 0.01$ and the momentum $\alpha = 0.9$; both terms were held constant throughout adaptation. The presentation of all of the patterns in the training data set is termed a *trial*; network weights and unit biases were updated after the presentation of each pattern during a trial.

The training data set for this problem was generated automatically by the image segmentation system. This training data set consists of approximately 4,000 ten element (feature) vectors (each vector describes one pixel); each vector is labeled as belonging to one of the 3 regions of interest in the image. The training data set was constructed from one entire training image, and is composed of vectors statistically representative of the pixels in each of the 3 regions of interest in that image.

All of the classifiers tested in this study were adapted from the same training data set. The connectionist classifier was defined to be converged for this problem before it was tested. Network convergence is determined from the results of two separate tests. In the first test, the difference between the network output and the target output averaged over the entire training data set has to reach a minimum. In the second test, the performance of the network in classifying the training data set is measured, and the number of misclassifications made by the network has to reach a minimum. Actual network performance in classifying a pattern is measured after post-processing of the output vector. The real outputs of each unit in the output layer are assigned the values of 0 or 1 by application of a 0.5 decision threshold. In our binary encoding scheme, the output vector should have only one element with the value 1; that element corresponds to one of the 3 classes. If the network produces an output vector with either more than one element with the value 1 or all elements with the value 0, the pattern generating that output is considered rejected. For the test problem in this study, all of the classifiers were set to reject patterns in the test data samples. All of the statistical classifiers had a rejection threshold set to 0.03.

## 3    Results

The performance of each of the classifiers (connectionist, Gaussian maximum likelihood, and linear, quadratic, and cubic polynomial) was measured on the training data set and test data representing 3 entire images taken from the series of combustion chamber images. One of those images, labeled Image 1, is the image from which the training data set was constructed. The performance of all of the classifiers is summarized in Table 1.

Although all of the classifiers were able to classify the training data set with comparably few misclassifications, the Gaussian maximum likelihood classifier and the quadratic polynomial classifier were unable to perform on any of the 3 entire test images. The connectionist classifier was the only alternative tested in this study to deliver acceptable performance on all 3 test images; the connectionist classifier had lower error rates on the test images than it delivered on the training data sample. Both the linear polynomial and cubic polynomial classifiers performed acceptably on the test Image 2, but then both exhibited high error rates on the other two test images. For this image segmentation problem, only the connectionist method generalized from the training data set to a solution with acceptable performance.

In Figure 3, the results from pixel classification performed by the connectionist and polynomial classifiers on all 3 test images are portrayed as segmented images. The actual test images are included at the left of the figure.

## 4    Conclusions

Our results demonstrate the feasibility of the application of a connectionist decision analysis method to the solution of a "real world" image segmentation problem. The

| Data Set | Connectionist Classifier | | Polynomial Classifier | | | Gaussian Classifier | |
|---|---|---|---|---|---|---|---|
| | Error[a] | Reject[b] | Degree | Error | Reject | Error | Reject |
| Training Data | 10.40% | 1.64% | 1 | 11.25% | 1.62% | 12.84% | 0.12% |
| | | | 2 | 9.61% | 1.41% | | |
| | | | 3 | 8.13% | 1.05% | | |
| Image 1[c] | 8.84% | 1.72% | 1 | 41.70% | 4.63% | 94.27% | 0.00% |
| | | | 2 | 57.55% | 3.66% | | |
| | | | 3 | 25.86% | 0.28% | | |
| Image 2 | 5.82% | 1.53% | 1 | 12.01% | 2.00% | 69.09% | 0.01% |
| | | | 2 | 68.01% | 0.58% | | |
| | | | 3 | 4.68% | 0.26% | | |
| Image 3 | 6.31% | 1.63% | 1 | 19.68% | 5.43% | 88.35% | 0.00% |
| | | | 2 | 45.89% | 1.41% | | |
| | | | 3 | 25.75% | 0.28% | | |

[a]Percent misclassifications for all patterns.

[b]Percent of all patterns rejected.

[c]Image from which training data set was taken.

**Table 1:** A summary of the performance of the classifiers.

inclusion of a connectionist classifier in our supervised segmentation system will allow us to meet our performance requirements under real world problem constraints.

Although the application of connectionism to the solution of real time machine vision problems represents a new processing method, our solution strategy has remained consistent with the decision analysis paradigm. Our connectionist classifiers are derived solely from the quantitative characteristics of the problem data; our connectionist architecture thus remains simple and need not be re-designed according to qualitative characteristics of each specific problem to which it will be applied. Our connectionist architecture is independent of the image size; we have applied the identical architecture successfully to images which range in size from 200 pixels by 200 pixels to 512 pixels by 512 pixels [9]. In most research to date in which neural networks are applied to machine vision, entire images explicitly are mapped to networks by making each pixel in an image correspond to a different unit in a network layer (see [10,11] for examples). This "pixel map" representation makes scaling up to larger image sizes from the idealized "toy" research images a significant problem.

Most statistical pattern classification methods require that problem data satisfy the assumptions of statistical models; unfortunately, real world problem data are complex and of variable quality and thus rarely can be used to guide the choice of an appropriate method for the solution of a particular problem *a priori*. For the image segmentation problem reported in this study, our classifier performance results show that the problem data actually did not satisfy the assumptions behind the statistical models underlying the Gaussian maximum likelihood classifier or the polynomial

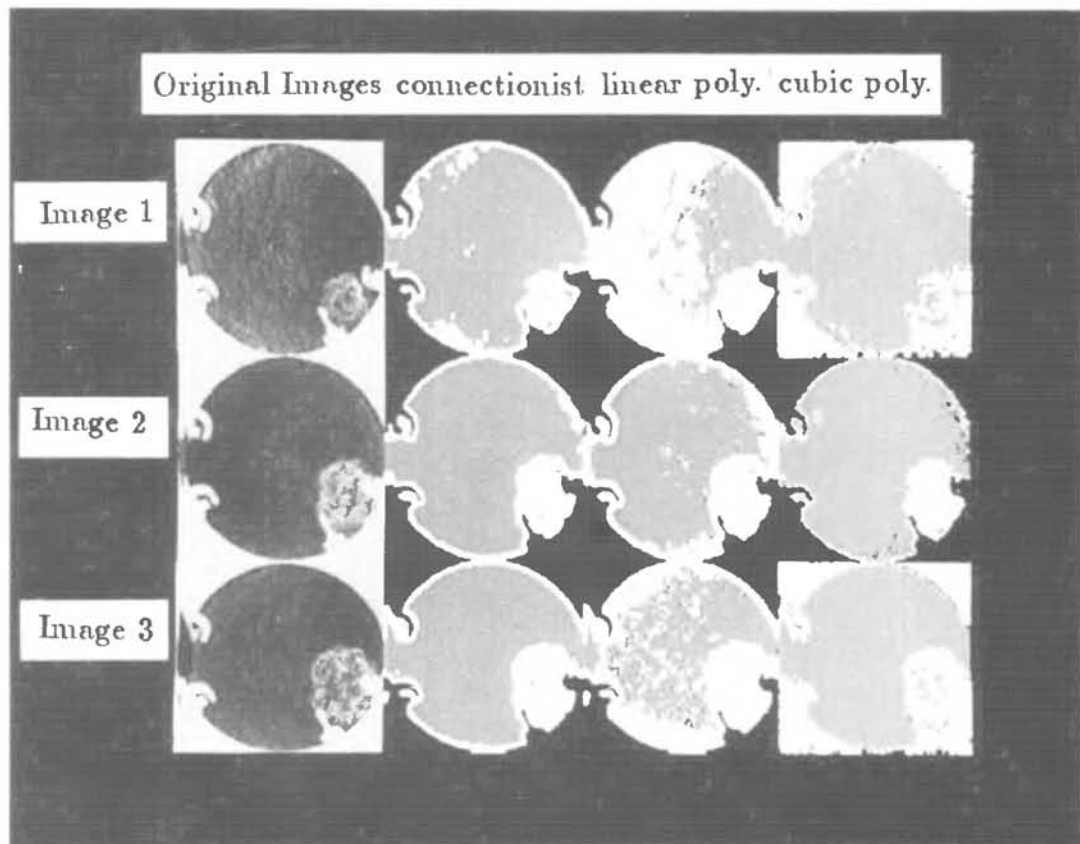

**Figure 3:** The grey levels assigned to each region are: Black — cylinder, Light Grey — uninflamed gas, Grey — flamed gas. Original images are at the left of the figure.

classifiers. It appears that the Gaussian model least fits our problem data, the polynomial classifiers provide a slightly better fit, and the connectionist method provides the fit required for the solution of the problem. It is also notable that all the alternative methods in this study could be adapted to perform acceptably on the training data set, but extensive testing on several different entire images was required in order to demonstrate the true performance of the alternative methods on the actual problem, rather than just on the training data set.

These results show that a connectionist method is a viable choice for a system such as ours which requires a simple architecture readily implemented in hardware, the flexibility to handle complex problems described by large amounts of data, and the robustness to not require problem data to meet many model assumptions *a priori*.

# References

[1] R. O. Duda and P. E. Hart. *Pattern Classification and Scene Analysis.* Wiley, New York, 1973.

[2] W. E. Blanz, J. L. C. Sanz, and D. Petkovic. Control-free low-level image segmentation: Theory, architecture,and experimentation. In J. L. C. Sanz, editor, *Advances of Machine Vision, Applications and Architectures*, Springer-Verlag, 1988.

[3] B. Straub and W. E. Blanz. Combined decision theoretic and syntactic approach to image segmentation. *Machine Vision and Applications*, 2(1):17–30, 1989.

[4] Sheri L. Gish and W. E. Blanz. *Comparing a Connectionist Trainable Classifier with Classical Statistical Decision Analysis Methods.* Research Report RJ 6891 (65717), IBM, June 1989.

[5] W. E. Blanz, B. Shung, C. Cox, W. Greiner, B. Dom, and D. Petković. *Design and implementation of a low level image segmentation architecture – LISA.* Research Report RJ 7194 (67673), IBM, December 1989.

[6] W. E. Blanz. Non-parametric feature selection for multiple class processes. In *Proc. 9th Int. Conf. Pattern Recognition*, Rome, Italy, Nov. 14–17 1988.

[7] David E. Rumelhart, James L. McClelland, et al. *Parallel Distributed Processing.* MIT Press, Cambridge, Massachusetts, 1986.

[8] David E. Rumelhart, Geoffrey E. Hinton, and Ronald J. Williams. Learning internal representations by error propagation. In David E. Rumelhart, James L. McClelland, et al., editors, *Parallel Distributed Processing*, chapter 8, MIT Press, Cambridge, Massachusetts, 1986.

[9] W. E. Blanz and Sheri L. Gish. *A Connectionist Classifier Architecture Applied To Image Segmentation.* Research Report RJ 7193 (67672), IBM, December 1989.

[10] K. Fukushima, S. Miyake, and T. Ito. Neocognitron: a neural network model for a mechanism of visual pattern recognition. *IEEE Transactions on Systems, Man, and Cybernetics*, SMC-13(5):826–834, 1983.

[11] Y. Hirai. A model of human associative processor. *IEEE Transactions on Systems, Man, and Cybernetics*, SMC-13(5):851–857, 1983.
